# Prediction on a Graph with a Perceptron

**Mark Herbster,   Massimiliano Pontil**
Department of Computer Science
University College London
Gower Street, London WC1E 6BT, England, UK
{*m.herbster, m.pontil*}*@cs.ucl.ac.uk*

## Abstract

We study the problem of online prediction of a noisy labeling of a graph with the perceptron. We address both label noise and concept noise. Graph learning is framed as an instance of prediction on a finite set. To treat label noise we show that the hinge loss bounds derived by Gentile [1] for online perceptron learning can be transformed to relative mistake bounds with an optimal leading constant when applied to prediction on a finite set. These bounds depend crucially on the norm of the learned concept. Often the norm of a concept can vary dramatically with only small perturbations in a labeling. We analyze a simple transformation that stabilizes the norm under perturbations. We derive an upper bound that depends only on natural properties of the graph – the graph diameter and the cut size of a partitioning of the graph – which are only indirectly dependent on the size of the graph. The impossibility of such bounds for the graph geodesic nearest neighbors algorithm will be demonstrated.

## 1   Introduction

We study the problem of robust online learning over a graph. Consider the following game for predicting the labeling of a graph. *Nature* presents a vertex $\mathbf{v}_{i_1}$; the *learner* predicts the label of the vertex $\hat{y}_1 \in \{-1, 1\}$; *nature* presents a label $y_1$; *nature* presents a vertex $\mathbf{v}_{i_2}$; the *learner* predicts $\hat{y}_2$; and so forth. The learner's goal is minimize the total number of mistakes ($|\{t : \hat{y}_t \neq y_t\}|$). If nature is adversarial, the learner will always mispredict; but if nature is regular or simple, there is hope that a learner may make only a few mispredictions. Thus, a methodological goal is to give learners whose total mispredictions can be bounded relative to the "complexity" of nature's labeling. In this paper, we consider the *cut size* as a measure of the complexity of a graph's labeling, where the size of the cut is the number of edges between disagreeing labels. We will give bounds which depend on the cut size and the diameter of the graph and thus do not directly depend on the size of the graph.

The problem of learning a labeling of a graph is a natural problem in the online learning setting, as well as a foundational technique for a variety of semi-supervised learning methods [2, 3, 4, 5, 6]. For example, in the online setting, consider a system which serves advertisements on web pages. The web pages may be identified with the vertices of a graph and the edges as links between pages. The online prediction problem is then that, at a given time $t$ the system may receive a request to serve an advertisement on a particular web page. For simplicity, we assume that there are two alternatives to be served: either advertisement "A" or advertisement "B". The system then interprets the feedback as the label and then may use this information in responding to the next request to predict an advertisement for a requested web page.

### 1.1   Related work

There is a well-developed literature regarding learning on the graph. The early work of Blum and Chawla [2] presented an algorithm which explicitly finds min-cuts of the label set. Bounds have been

**Input:** $\{(\mathbf{v}_{i_t}, y_t)\}_{t=1}^{\ell} \subseteq \mathcal{V}_\mathbf{M} \times \{-1, 1\}$.
**Initialization:** $\mathbf{w}_1 = \mathbf{0}$; $\mathcal{M}_A = \emptyset$.
**for** $t = 1, \ldots, \ell$ **do**
  **Predict:** receive $\mathbf{v}_{i_t}$
  $\hat{y}_t = \text{sign}(\mathbf{e}_{i_t}^\top \mathbf{w}_t)$
  **Update:** receive $y_t$
  **if** $\hat{y}_t = y_t$ **then**
    $\mathbf{w}_{t+1} = \mathbf{w}_t$
  **else**
    $\mathbf{w}_{t+1} = \mathbf{w}_t + y_t \mathbf{v}_{i_t}$
    $\mathcal{M}_A = \mathcal{M}_A \cup \{t\}$
**end**

Figure 1: Perceptron on set $\mathcal{V}_\mathbf{M}$.

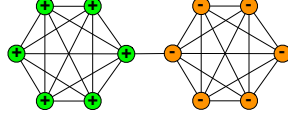

Figure 2: Barbell

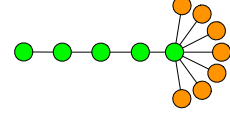

Figure 4: Flower

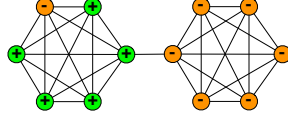

Figure 3: Barbell with concept noise

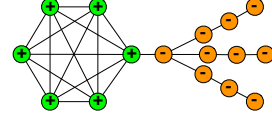

Figure 5: Octopus

proven previously with smooth loss functions [6, 7] in a batch setting. Kernels on graph labelings were introduced in [3, 5]. This current work builds upon our work in [8]. There it was shown that, given a fixed labeling of a graph, the number of mistakes made by an algorithm similar to the kernel perceptron [9] with a kernel that was the pseudoinverse of the graph Laplacian, could be bounded by the quantity [8, Theorems 3.2, 4.1, and 4.2]

$$4\Phi_\mathbf{G}(\mathbf{u}) D_\mathbf{G} \text{bal}(\mathbf{u}). \tag{1}$$

Here $\mathbf{u} \in \{-1, 1\}^n$ is a binary vector defining the labeling of the graph, $\Phi_\mathbf{G}(\mathbf{u})$ is the cut size[1] defined as $\Phi_\mathbf{G}(\mathbf{u}) := |\{(i, j) \in E(\mathbf{G}) : u_i \neq u_j\}|$, that is, the number of edges between positive and negative labels, $D_\mathbf{G}$ is the diameter of the graph and $\text{bal}(\mathbf{u}) := \left(1 - \frac{1}{n}|\sum u_i|\right)^{-2}$ measures the *label balance*. This bound is interesting in that the mistakes of the algorithm could be bounded in terms of simple properties of a labeled graph. However, there are a variety of shortcomings in this result. First, we observe that the bound above assumed a fixed labeling of the graph. In practice, the online data sequence could contain multiple labels for a single vertex; this is the problem of *label noise*. Second, for an unbalanced set of labels the bound is vacuous, for example, if $\mathbf{u} = \{1, 1, \ldots, 1, -1\} \in \mathbb{R}^n$ then $\text{bal}(\mathbf{u}) = n^2$. Third, consider the prototypical easy instance for the algorithm of two dense clusters connected by a few edges, for instance, two $m$-cliques connected by a single edge (a barbell graph, see Figure 2). If each clique is labeled with distinct labels then we have that $4\Phi_\mathbf{G}(\mathbf{u}) D_\mathbf{G} \text{bal}(\mathbf{u}) = 4 \times 1 \times 3 \times 1 = 12$, which is independent of $m$. Now suppose that, say, the first clique contains one vertex which is labeled as the second clique (see Figure 3). Previously $\Phi_\mathbf{G}(\mathbf{u}) = 1$, but now $\Phi_\mathbf{G}(\mathbf{u}) = m$ and the bound is vacuous. This is the problem of *concept noise*; in this example, a $\Theta(1)$ perturbation of labeling increases the bound multiplicatively by $\Theta(m)$.

## 1.2 Overview

A first aim of this paper is to improve upon the bounds in [8], particularly, to address the three problems of label balance, label noise, and concept noise as discussed above. For this purpose, we apply the well-known kernel perceptron [9] to the problem of online learning on the graph. We discuss the background material for this problem in section 2, where we also show that the bounds of [1] can be specialized to relative mistake bounds when applied to, for example, prediction on the graph. A second important aim of this paper is to interpret the mistake bounds by an explanation in terms of high level graph properties. Hence, in section 3, we refine a diameter based bound of [8, Theorem 4.2] to a sharper bound based on the "resistance distance" [10] on a weighted graph; which we then closely match with a lower bound. In section 4, we introduce a kernel which is a simple augmentation of the pseudoinverse of the graph Laplacian and then prove a theorem on the performance of the perceptron with this kernel which solves the three problems above. We conclude in section 5, with a discussion comparing the mistake bounds for prediction on the graph with the *halving* algorithm [11] and the $k$-nearest neighbors algorithm.

## 2 Preliminaries

In this section, we describe our setup for Hilbert spaces on finite sets and its specification to the graph case. We then recall a result of Gentile [1] on prediction with the perceptron and discuss a special case in which *relative* 0–1 loss (mistake) bounds are obtainable.

## 2.1 Hilbert spaces of functions defined on a finite set

We denote matrices by capital bold letters and vectors by small bold case letters. So $\mathbf{M}$ denotes the $n \times n$ matrix $(M_{ij})_{i,j=1}^n$ and $\mathbf{w}$ the $n$-dimensional vector $(w_i)_{i=1}^n$. The identity matrix is denoted by $\mathbf{I}$. We also let $\mathbf{0}$ and $\mathbf{1}$ be the $n$-dimensional vectors all of whose components equal to zero and one respectively, and $\mathbf{e}_i$ the $i$-th coordinate vector of $\mathbb{R}^n$. Let $\mathbb{N}$ be the set of natural numbers and $\mathbb{N}_\ell := \{1, \ldots, \ell\}$. We denote a generic Hilbert space with $\mathcal{H}$. We identify $V := \mathbb{N}_n$ as the indices of a set of $n$ objects, e.g. the vertices of a graph. A vector $\mathbf{w} \in \mathbb{R}^n$ can alternatively be seen as a function $f : V \to \mathbb{R}$ such that $f(i) = w_i$, $i \in V$. However, for simplicity we will use the notation $\mathbf{w}$ to denote both a vector in $\mathbb{R}^n$ or the above function.

A symmetric positive semidefinite matrix $\mathbf{M}$ induces a semi-inner product on $\mathbb{R}^n$ which is defined as $\langle \mathbf{u}, \mathbf{w} \rangle_{\mathbf{M}} := \mathbf{u}^\top \mathbf{M} \mathbf{w}$, where "$\top$" denotes transposition. The reproducing kernel [12] associated with the above semi-inner product is $\mathbf{K} = \mathbf{M}^+$, where "$+$" denotes pseudoinverse. We also define the *coordinate spanning set*

$$\mathcal{V}_{\mathbf{M}} := \{\mathbf{v}_i := \mathbf{M}^+ \mathbf{e}_i : i = 1, \ldots, n\} \tag{2}$$

and let $\mathcal{H}(\mathbf{M}) := \mathrm{span}(\mathcal{V}_{\mathbf{M}})$. The restriction of the semi-inner product $\langle \cdot, \cdot \rangle_{\mathbf{M}}$ to $\mathcal{H}(\mathbf{M})$ is an inner product on $\mathcal{H}(\mathbf{M})$. The set $\mathcal{V}_{\mathbf{M}}$ acts as "coordinates" for $\mathcal{H}(\mathbf{M})$, that is, if $\mathbf{w} \in \mathcal{H}(\mathbf{M})$ we have

$$w_i = \mathbf{e}_i^\top \mathbf{M}^+ \mathbf{M} \mathbf{w} = \mathbf{v}_i^\top \mathbf{M} \mathbf{w} = \langle \mathbf{v}_i, \mathbf{w} \rangle_{\mathbf{M}}, \tag{3}$$

although the vectors $\{\mathbf{v}_1, \ldots, \mathbf{v}_n\}$ are not necessarily normalized and are linearly independent only if $\mathbf{M}$ is positive definite. We note that equation (3) is simply the reproducing kernel property [12] for kernel $\mathbf{M}^+$.

When $V$ indexes the vertices of an undirected graph $\mathcal{G}$, a natural norm to use is that induced by the graph Laplacian. We explain this in detail now. Let $\mathbf{A}$ be the $n \times n$ symmetric weight matrix of the graph such that $A_{ij} \geq 0$, and define the edge set $E(\mathcal{G}) := \{(i,j) : 0 < A_{ij}, i < j\}$. The distance matrix $\mathbf{\Delta}$ associated with $\mathcal{G}$ is the per-element inverse of the weight matrix, that is, $\Delta_{ij} = \frac{1}{A_{ij}}$ ($\mathbf{\Delta}$ may have $+\infty$ as a matrix element). The graph Laplacian $\mathbf{G}$ is the $n \times n$ matrix defined as $\mathbf{G} := \mathbf{D} - \mathbf{A}$ where $\mathbf{D} = \mathrm{diag}(d_1, \ldots, d_n)$ and $d_i$ is the *weighted* degree of vertex $i$, $d_i = \sum_{j=1}^n A_{ij}$. The Laplacian is positive semidefinite and induces the semi-norm

$$\|\mathbf{w}\|_{\mathbf{G}}^2 := \mathbf{w}^\top \mathbf{G} \mathbf{w} = \sum_{(i,j) \in E(\mathbf{G})} A_{ij}(w_i - w_j)^2. \tag{4}$$

When the graph is connected, it follows from equation (4) that the null space of $\mathbf{G}$ is spanned by the constant vector $\mathbf{1}$ only. In this paper, we always assume that the graph $\mathcal{G}$ is connected. Where it is not ambiguous, we use the notation $\mathbf{G}$ to denote both the graph $\mathcal{G}$ and the graph Laplacian.

## 2.2 Online prediction of functions on a finite set with the perceptron

Gentile [1] bounded the performance of the perceptron algorithm on nonseparable data with the linear hinge loss. Here, we apply his result to study the problem of prediction on a finite set with the perceptron (see Figure 1). In this case, the inputs are the coordinates in the set $\mathcal{V}_{\mathbf{M}} \subset \mathcal{H}(\mathbf{M})$ defined above. We additionally assume that matrix $\mathbf{M}$ is *positive definite* (not just positive semidefinite as in the previous subsection). This assumption, along with the fact that the inputs are coordinates, enables us to upper bound the hinge loss and hence we may give a *relative* mistake bound in terms of the complete set of base classifiers $\{-1, 1\}^n$.

**Theorem 2.1.** *Let $\mathbf{M}$ be a symmetric positive definite matrix. If $\{(\mathbf{v}_{i_t}, y_t)\}_{t=1}^\ell \subseteq \mathcal{V}_{\mathbf{M}} \times \{-1, 1\}$ is a sequence of examples, $\mathcal{M}_A$ denotes the set of trials in which the perceptron algorithm predicted incorrectly and $X = \max_{t \in \mathcal{M}_A} \|\mathbf{v}_{i_t}\|_{\mathbf{M}}$, then the cumulative number of mistakes $|\mathcal{M}_A|$ of the algorithm is bounded by*

$$|\mathcal{M}_A| \leq 2|\mathcal{M}_A \cap \mathcal{M}_{\mathbf{u}}| + \frac{\|\mathbf{u}\|_{\mathbf{M}}^2 X^2}{2} + \sqrt{2|\mathcal{M}_A \cap \mathcal{M}_{\mathbf{u}}|\|\mathbf{u}\|_{\mathbf{M}}^2 X^2 + \frac{\|\mathbf{u}\|_{\mathbf{M}}^4 X^4}{4}} \tag{5}$$

*for all $\mathbf{u} \in \{-1, 1\}^n$, where $\mathcal{M}_{\mathbf{u}} = \{t \in \mathbb{N}_\ell : u_{i_t} \neq y_t\}$. In particular, if $|\mathcal{M}_{\mathbf{u}}| = 0$ then*

$$|\mathcal{M}_A| \leq \|\mathbf{u}\|_{\mathbf{M}}^2 X^2.$$

*Proof.* This bound follows directly from [1, Theorem 8] with $p = 2$, $\gamma = 1$, and $\mathbf{w}_1 = 0$. Since $\mathbf{M}$ is assumed to be symmetric positive definite, it follows that $\{-1, 1\}^n \subset \mathcal{H}(\mathbf{M})$. Thus, the hinge loss $L_{\mathbf{u},t} := \max(0, 1 - y_t \langle \mathbf{u}, \mathbf{v}_{i_t} \rangle_{\mathbf{M}})$ of any classifier $\mathbf{u} \in \{-1, 1\}^n$ with any example $(\mathbf{v}_{i_t}, y_t)$ is either 0 or 2, since $|\langle \mathbf{u}, \mathbf{v}_{i_t} \rangle_{\mathbf{M}}| = 1$ by equation (3). This allows us to bound the hinge loss term of [1, Theorem 8] directly with mistakes. $\square$

We emphasize that our hypothesis on $\mathbf{M}$ does not imply linear separability since multiple instances of an input vector in the training sequence may have distinct target labels. Moreover, we note that, for deterministic prediction the constant 2 in the first term of the right hand side of equation (5) is optimal for an online algorithm as a mistake may be forced on every trial.

## 3   Interpretation of the space $\mathcal{H}(\mathbf{G})$

The bound for prediction on a finite set in equation (5) involves two quantities, namely the squared norm of a classifier $\mathbf{u} \in \{-1, 1\}^n$ and the maximum of the squared norms of the coordinates $\mathbf{v} \in \mathcal{V}_{\mathbf{M}}$. In the case of prediction on the graph, recall from equation (4) that $\|\mathbf{u}\|_{\mathbf{G}}^2 := \mathbf{u}^\top \mathbf{G} \mathbf{u} = \sum_{(i,j) \in E(\mathbf{G})} A_{ij}(u_i - u_j)^2$. Therefore, we may identify this semi-norm with the *weighted cut size*

$$\Phi_{\mathbf{G}}(\mathbf{u}) := \frac{1}{4} \|\mathbf{u}\|_{\mathbf{G}}^2 \tag{6}$$

of the labeling induced when $\mathbf{u} \in \{-1, 1\}^n$. In particular, with boolean weighted edges ($A_{ij} \in \{0, 1\}$) the cut simply counts the number of edges spanning disagreeing labels.

The norm $\|\mathbf{v} - \mathbf{w}\|_{\mathbf{G}}$ is a *metric* distance for $\mathbf{v}, \mathbf{w} \in \text{span}(\mathcal{V}_{\mathbf{G}})$ however, surprisingly, the *square* of the norm $\|\mathbf{v}_p - \mathbf{v}_q\|_{\mathbf{G}}^2$ when restricted to graph coordinates $\mathbf{v}_p, \mathbf{v}_q \in \mathcal{V}_{\mathbf{G}}$ is also a metric known as the *resistance distance* [10],

$$r_{\mathbf{G}}(p, q) := (\mathbf{e}_p - \mathbf{e}_q)^\top \mathbf{G}^+ (\mathbf{e}_p - \mathbf{e}_q) = \|\mathbf{v}_p - \mathbf{v}_q\|_{\mathbf{G}}^2. \tag{7}$$

It is interesting to note that the resistance distance between vertex $p$ and vertex $q$ is the *effective resistance* between vertices $p$ and $q$, where the graph is the circuit and edge $(i, j)$ is a resistor with the resistance $\Delta_{ij} = A_{ij}^{-1}$.

As we shall see, our bounds in section 4 depend on $\|\mathbf{v}_p\|_{\mathbf{G}}^2 = \|\mathbf{v}_p - \mathbf{0}\|_{\mathbf{G}}^2$. Formally, this is not an effective resistance between vertex $p$ and another vertex "$\mathbf{0}$". The vector $\mathbf{0}$, informally however, is the center of the graph as $\mathbf{0} = \frac{\sum_{\mathbf{v} \in \mathcal{V}_{\mathbf{G}}} \mathbf{v}}{|\mathcal{V}_{\mathbf{G}}|}$, since $\mathbf{1}$ is in the null space of $\mathcal{H}(\mathbf{G})$. In the following, we further characterize $\|\mathbf{v}_p\|_{\mathbf{G}}^2$. First, we observe qualitatively that the more interconnected the graph the smaller the term $\|\mathbf{v}_p\|_{\mathbf{G}}^2$ (Corollary 3.1). Second, in Theorem 3.2 we quantitatively upper bound $\|\mathbf{v}_p\|_{\mathbf{G}}^2$ by the average (over $q$) of the effective resistance between vertex $p$ and each vertex $q$ in the graph (including $q = p$), which in turn may be upper bounded by the eccentricity of $p$. We proceed with the following useful lemma and theorem, as a basis for our later results.

**Lemma 3.1.** *Let* $\mathbf{x} \in \mathcal{H}$ *then* $\|\mathbf{x}\|^{-2} = \min_{\mathbf{w} \in \mathcal{H}} \{\|\mathbf{w}\|^2 : \langle \mathbf{w}, \mathbf{x} \rangle = 1\}$.

The proof is straightforward and we do not elaborate on the details.

**Theorem 3.1.** *If* $\mathbf{M}$ *and* $\mathbf{M}'$ *are symmetric positive semidefinite matrices with* $\text{span}(\mathcal{V}_{\mathbf{M}}) = \text{span}(\mathcal{V}_{\mathbf{M}'})$ *and, for every* $\mathbf{w} \in \text{span}(\mathcal{V}_{\mathbf{M}})$, $\|\mathbf{w}\|_{\mathbf{M}}^2 \leq \|\mathbf{w}\|_{\mathbf{M}'}^2$ *then*

$$\left\|\sum_{i=1}^n a_i \mathbf{v}_i\right\|_{\mathbf{M}}^2 \geq \left\|\sum_{i=1}^n a_i \mathbf{v}_i'\right\|_{\mathbf{M}'}^2,$$

*where* $\mathbf{v}_i \in \mathcal{V}_{\mathbf{M}}$, $\mathbf{v}_i' \in \mathcal{V}_{\mathbf{M}'}$ *and* $\mathbf{a} \in \mathbb{R}^n$.

*Proof.* Let $\mathbf{x} = \sum_{i=1}^n a_i \mathbf{v}_i$ and $\mathbf{x}' = \sum_{i=1}^n a_i \mathbf{v}_i'$ then

$$\|\mathbf{x}\|_{\mathbf{M}}^{-2} = \left\|\frac{\mathbf{x}}{\|\mathbf{x}\|_{\mathbf{M}}^2}\right\|_{\mathbf{M}}^2 \leq \left\|\frac{\mathbf{x}'}{\|\mathbf{x}'\|_{\mathbf{M}'}^2}\right\|_{\mathbf{M}}^2 \leq \left\|\frac{\mathbf{x}'}{\|\mathbf{x}'\|_{\mathbf{M}'}^2}\right\|_{\mathbf{M}'}^2 = \|\mathbf{x}'\|_{\mathbf{M}'}^{-2},$$

where the first inequality follows since $\langle \frac{\mathbf{x}'}{\|\mathbf{x}'\|_{\mathbf{M}'}^2}, \mathbf{x} \rangle_{\mathbf{M}} = 1$, hence $\frac{\mathbf{x}'}{\|\mathbf{x}'\|_{\mathbf{M}'}^2}$ is a feasible solution to the minimization problem in the right hand side of Lemma 3.1. While the second one follows immediately from the assumption that $\|\mathbf{w}\|_{\mathbf{M}}^2 \leq \|\mathbf{w}\|_{\mathbf{M}'}^2$. $\square$

As a corollary to the above theorem we have the following when $\mathbf{M}$ is a graph Laplacian.

**Corollary 3.1.** *Given connected graphs $\mathbf{G}$ and $\mathbf{G}'$ with distance matrices $\mathbf{\Delta}$ and $\mathbf{\Delta}'$ such that $\Delta_{ij} \leq \Delta'_{ij}$ then for all $p, q \in V$, we have that $\|\mathbf{v}_p\|_{\mathbf{G}}^2 \leq \|\mathbf{v}'_p\|_{\mathbf{G}'}^2$ and $r_{\mathbf{G}}(p, q) \leq r_{\mathbf{G}'}(p, q)$.*

The first inequality in the above corollary demonstrates that $\|\mathbf{v}_p\|_G^2$ is nonincreasing in a graph that is strictly more connected. The second inequality is the well-known Rayleigh's *monotonicity law* which states that if any resistance in a circuit is decreased then the effective resistance between any two points cannot increase.

We define the geodesic distance between vertices $p, q \in V$ to be $d_{\mathbf{G}}(p, q) := \min |P(p, q)|$ where the minimum is taken with respect to all paths $\mathcal{P}(p, q)$ from $p$ to $q$, with the path length defined as $|\mathcal{P}(p, q)| := \sum_{(i,j) \in E(\mathcal{P}(p,q))} \Delta_{ij}$. The eccentricity $d_{\mathbf{G}}(p)$ of a vertex $p \in V$ is the geodesic distance on the graph between $p$ and the furthest vertex on the graph to $p$, that is, $d_{\mathbf{G}}(p) = \max_{q \in V} d_{\mathbf{G}}(p, q) \leq D_{\mathbf{G}}$, and $D_{\mathbf{G}}$ is the (geodesic) diameter of the graph, $D_{\mathbf{G}} := \max_{p \in V} d_{\mathbf{G}}(p)$. A graph $\mathcal{G}$ is connected when $D_{\mathbf{G}} < \infty$. A tree is an $n$-vertex connected graph with $n - 1$ edges. The following lemma, a well known result (see e.g. [10]), establishes that the resistance distance can be be equated with the geodesic distance when the graph is a tree.

**Lemma 3.2.** *If the graph $\mathcal{T}$ is a tree with graph Laplacian $\mathbf{T}$ then $r_{\mathbf{T}}(p, q) = d_{\mathbf{T}}(p, q)$.*

The next theorem provides a quantitative relationship between $\|\mathbf{v}_p\|_{\mathbf{G}}^2$ and two measures of the connectivity of vertex $p$, namely its eccentricity and the mean of the effective resistances between vertex $p$ and each vertex on the graph.

**Theorem 3.2.** *If $\mathbf{G}$ is a connected graph then*

$$\|\mathbf{v}_p\|_{\mathbf{G}}^2 \leq \frac{1}{n} \sum_{q=1}^{n} r_{\mathbf{G}}(p, q) \leq d_{\mathbf{G}}(p). \tag{8}$$

*Proof.* Recall that $r_{\mathbf{G}}(p, q) = \|\mathbf{v}_p - \mathbf{v}_q\|_{\mathbf{G}}^2$ (see equation (7)) and use $\sum_{q=1}^{n} \mathbf{v}_q = \mathbf{0}$ to obtain that $\frac{1}{n} \sum_{q=1}^{n} \|\mathbf{v}_p - \mathbf{v}_q\|_{\mathbf{G}}^2 = \mathbf{v}_p^\top \mathbf{G} \mathbf{v}_p + \frac{1}{n} \sum_{q=1}^{n} \mathbf{v}_q^\top \mathbf{G} \mathbf{v}_q$ which implies the left inequality in (8). Next, by Corollary 3.1, if $\mathbf{T}$ is the Laplacian of a tree $\mathbf{T} \subseteq \mathbf{G}$ then $r_{\mathbf{G}}(p, q) \leq r_{\mathbf{T}}(p, q)$ for $p, q \in V$. Therefore, from Lemma 3.2 we conclude that $r_{\mathbf{G}}(p, q) \leq d_{\mathbf{T}}(p, q)$. Moreover, since $\mathbf{T} \subseteq \mathbf{G}$ can be any tree, we have that $r_{\mathbf{G}}(p, q) \leq \min_{\mathbf{T}} d_{\mathbf{T}}(p, q)$ where the minimum is over all trees $\mathbf{T} \subseteq \mathbf{G}$. Since the geodesic path from $p$ to $q$ is necessarily contained in some tree $\mathbf{T} \subseteq \mathbf{G}$ it follows that $\min_{\mathbf{T}} d_{\mathbf{T}}(p, q) = d_{\mathbf{G}}(p, q)$ and, so, $r_{\mathbf{G}}(p, q) \leq d_{\mathbf{G}}(p, q)$. Now the theorem follows by maximizing $d_{\mathbf{G}}(p, q)$ over $q$ and the definition of $d_{\mathbf{G}}(p)$. $\square$

We identify the *resistance diameter* of a graph $\mathbf{G}$ as $R_{\mathbf{G}} := \max_{p, q \in V} r_{\mathbf{G}}(p, q)$; thus, from the previous theorem, we may also conclude that

$$\max_{p \in V} \|\mathbf{v}_p\|_{\mathbf{G}}^2 \leq R_{\mathbf{G}} \leq D_{\mathbf{G}}. \tag{9}$$

We complete this section by showing that there exists a family of graphs for which the above inequality is nearly tight. Specifically, we consider the "flower graph" (see Figure 4) obtained by connecting the first vertex of a chain with $p - 1$ vertices to the root vertex of an $m$-ary tree of depth one. We index the vertices of this graph so that vertices 1 to $p$ correspond to "stem vertices" and vertices $p + 1$ to $p + m$ to "petals". Clearly, this graph has diameter equal to $p$, hence our upper bound above establishes that $\|\mathbf{v}_1\|_{\mathbf{G}}^2 \leq p$. We now argue that as $m$ grows this bound is almost tight. From Lemma 3.1 we have that $\|\mathbf{v}_1\|_{\mathbf{G}}^{-2} = \min_{\mathbf{w} \in \mathcal{H}(\mathbf{G})} \{\|\mathbf{w}\|_{\mathbf{G}}^2 : \langle \mathbf{w}, \mathbf{v}_1 \rangle = 1\}$. We note that by symmetry, the solution $\hat{\mathbf{w}} = (\hat{w}_i : i \in \mathbb{N}_{p+m})$ to the problem above satisfies $\hat{w}_i = z$ if $i \geq p + 1$ since $\hat{\mathbf{w}}$ must take the same value on the petal vertices. Consequently, it follows that $\|\mathbf{v}_1\|_{\mathbf{G}}^{-2} = \min \left\{ m(z - w_p)^2 + \sum_{i=1}^{p-1} (w_i - w_{i+1})^2 : w_1 = 1, \sum_{i=1}^{p} w_i + mz = 0 \right\}$. We upper bound this minimum by choosing $w_i = \frac{p-i}{p-1}$ for $1 \leq i \leq p$. Thus, $w_1 = 1$ as it is required, $w_p = 0$ and we compute $z$ by the constraint set of the above minimization problem as $z = -\frac{p}{2m}$. A direct computation gives $\|\mathbf{v}_1\|_{\mathbf{G}}^{-2} \leq \frac{1}{(p-1)} + \frac{p^2}{4m}$ from which using a first order Taylor expansion it follows that $\|\mathbf{v}_1\|_{\mathbf{G}}^2 \geq (p-1) - \frac{(p-1)^2 p^2}{4m}$. Therefore, as $m \to \infty$ the upper bound on $\|\mathbf{v}_1\|_{\mathbf{G}}^2$ (equation (8)) for the flower graph is matched by a lower bound with a gap of 1.

# 4 Prediction on the graph

We define the following symmetric positive definite graph kernel,

$$\mathbf{K}_c^b := \mathbf{G}^+ + b\mathbf{1}\mathbf{1}^\top + c\mathbf{I}, \quad (0 < b,\ 0 \le c), \tag{10}$$

where $\mathbf{G}_c^b = (\mathbf{K}_c^b)^{-1}$ is the matrix of the associated Hilbert space $\mathcal{H}(\mathbf{G}_c^b)$. In Lemma 4.1 below we prove the needed properties of $\mathcal{H}(\mathbf{G}_c^b)$ as a necessary step for the bound in Theorem 4.2. As we shall see, these properties moderate the consequences of label imbalance and concept noise. To prove Lemma 4.1, we use the following theorem which is a special case of [12, Thm I, §I.6].

**Theorem 4.1.** *If $\mathbf{M}_1$ and $\mathbf{M}_2$ are $n \times n$ symmetric positive semidefinite matrices, and we set $\mathbf{M} := (\mathbf{M}_1^+ + \mathbf{M}_2^+)^+$ then $\|\mathbf{w}\|_{\mathbf{M}}^2 = \inf\{\|\mathbf{w}_1\|_{\mathbf{M}_1}^2 + \|\mathbf{w}_2\|_{\mathbf{M}_2}^2 : \mathbf{w}_i \in \mathcal{H}(\mathbf{M}_i),\ \mathbf{w}_1 + \mathbf{w}_2 = \mathbf{w}\}$ for every $\mathbf{w} \in \mathcal{H}(\mathbf{M})$.*

Next, we define $\beta_{\mathbf{u}} \in [0,1]$ as a measure of the balance of a labeling $\mathbf{u} \in \{-1,1\}^n$ as $\beta_{\mathbf{u}} := (\frac{1}{n}\sum_{i=1}^n u_i)^2$. Note that for a perfectly balanced labeling $\beta_{\mathbf{u}} = 0$, while $\beta_{\mathbf{u}} = 1$ for a perfectly unbalanced one.

**Lemma 4.1.** *Given a vertex $p$ with associated coordinates $\mathbf{v}_p \in \mathcal{V}_{\mathbf{G}}$ and $\mathbf{v}_p' \in \mathcal{V}_{\mathbf{G}_c^b}$ we have that*

$$\|\mathbf{v}_p'\|_{\mathbf{G}_c^b}^2 = \|\mathbf{v}_p\|_{\mathbf{G}}^2 + b + c. \tag{11}$$

*Moreover, if $\mathbf{u}, \mathbf{u}' \in \{-1,1\}^n$ and where $k := |\{i : u_i' \ne u_i\}|$ we have that*

$$\|\mathbf{u}'\|_{\mathbf{G}_c^b}^2 \le \|\mathbf{u}\|_{\mathbf{G}}^2 + \frac{\beta_{\mathbf{u}}}{b} + \frac{4k}{c}. \tag{12}$$

*Proof.* To prove equation (11) we recall equation (3) and note that $\|\mathbf{v}_p'\|_{\mathbf{G}_c^b}^2 = \langle \mathbf{v}_p', \mathbf{v}_p + b\mathbf{1} + c\mathbf{e}_p \rangle_{\mathbf{G}_c^b} = \langle \mathbf{v}_p', \mathbf{v}_p \rangle_{\mathbf{G}_c^b} + \langle \mathbf{v}_p', b\mathbf{1} + c\mathbf{e}_p \rangle_{\mathbf{G}_c^b} = \|\mathbf{v}_p\|_{\mathbf{G}}^2 + b + c.$

To prove inequality (12) we proceed in two steps. First, we show that

$$\|\mathbf{u}\|_{\mathbf{G}_0^b}^2 = \|\mathbf{u}\|_{\mathbf{G}}^2 + \frac{\beta_{\mathbf{u}}}{b}. \tag{13}$$

Indeed, we can uniquely decompose $\mathbf{u}$ as the sum of a vector in $\mathcal{H}(\mathbf{G})$ and one in $\mathcal{H}(\frac{\mathbf{1}\mathbf{1}^\top}{n^2 b})$ as $\mathbf{u} = (\mathbf{u} - \mathbf{1}\frac{1}{n}\sum_{i=1}^n u_i) + \mathbf{1}\frac{1}{n}\sum_{i=1}^n u_i$. Therefore, by Theorem 4.1 we conclude that $\|\mathbf{u}\|_{\mathbf{G}_0^b}^2 = \|\mathbf{u} - \sqrt{\beta_{\mathbf{u}}}\mathbf{1}\|_{\mathbf{G}}^2 + \|\sqrt{\beta_{\mathbf{u}}}\mathbf{1}\|_{\frac{\mathbf{1}\mathbf{1}^\top}{n^2 b}}^2 = \|\mathbf{u}\|_{\mathbf{G}}^2 + \frac{\beta_{\mathbf{u}}}{b}$, where $\|\mathbf{u} - \sqrt{\beta_{\mathbf{u}}}\mathbf{1}\|_{\mathbf{G}}^2 = \|\mathbf{u}\|_{\mathbf{G}}^2$ since $\mathbf{1} \in \mathcal{H}^\perp(\mathbf{G})$.
Second, we show, for any symmetric positive definite matrix $\mathbf{M}$, $\mathbf{u}, \mathbf{u}' \in \{-1,1\}^n$ and $c > 0$, that

$$\|\mathbf{u}'\|_{\mathbf{M}_c}^2 \le \|\mathbf{u}\|_{\mathbf{M}}^2 + \frac{4k}{c}, \tag{14}$$

where $\mathbf{M}_c := (\mathbf{M}^{-1} + c\mathbf{I})^{-1}$ and $k := |\{i : u_i' \ne u_i\}|$. To this end, we decompose $\mathbf{u}'$ as a sum of two elements of $\mathcal{H}(\mathbf{M})$ and $\mathcal{H}(\frac{1}{c}\mathbf{I})$ as $\mathbf{u}' = \mathbf{u} + (\mathbf{u}' - \mathbf{u})$ and observe that $\|\mathbf{u}' - \mathbf{u}\|_{\frac{1}{c}\mathbf{I}}^2 = \frac{4k}{c}$. By Theorem 4.1 it then follows that $\|\mathbf{u}'\|_{\mathbf{M}_c}^2 \le \|\mathbf{u}\|_{\mathbf{M}}^2 + \|\mathbf{u}' - \mathbf{u}\|_{\frac{1}{c}\mathbf{I}}^2 = \|\mathbf{u}\|_{\mathbf{M}}^2 + \frac{4k}{c}$. Now inequality (12) follows by combining equations (13) and (14) with $\mathbf{M} = \mathbf{G}_0^b$. $\square$

We can now state our relative mistake bound for online prediction on the graph.

**Theorem 4.2.** *Let $\mathbf{G}$ be a connected graph. If $\{(\mathbf{v}_{i_t}, y_t)\}_{t=1}^\ell \subseteq \mathcal{V}_{\mathbf{G}_c^b} \times \{-1,1\}$ is a sequence of examples and $\mathcal{M}_A$ denotes the set of trials in which the perceptron algorithm predicted incorrectly, then the cumulative number of mistakes $|\mathcal{M}_A|$ of the algorithm is bounded by*

$$|\mathcal{M}_A| \le 2|\mathcal{M}_A \cap \mathcal{M}_{\mathbf{u}}| + \frac{\mathcal{Z}}{2} + \sqrt{2|\mathcal{M}_A \cap \mathcal{M}_{\mathbf{u}}|\mathcal{Z} + \frac{\mathcal{Z}^2}{4}}, \tag{15}$$

*for all $\mathbf{u}, \mathbf{u}' \in \{-1,1\}^n$, where $k = |\{i : u_i' \ne u_i\}|$, $\beta_{\mathbf{u}'} = (\frac{1}{n}\sum_{i=1}^n u_i')^2$, $\mathcal{M}_{\mathbf{u}} = \{t \in \mathbb{N}_\ell : u_{i_t} \ne y_t\}$, and*

$$\mathcal{Z} = \left(4\Phi_{\mathbf{G}}(\mathbf{u}') + \frac{\beta_{\mathbf{u}'}}{b} + \frac{4k}{c}\right)\left(R_{\mathbf{G}} + b + c\right).$$

*In particular, if $b = 1$, $c = 0$, $k = 0$ and $|\mathcal{M}_{\mathbf{u}}| = 0$ then*

$$|\mathcal{M}_A| \le (4\Phi_{\mathbf{G}}(\mathbf{u}) + \beta_{\mathbf{u}})(R_{\mathbf{G}} + 1). \tag{16}$$

*Proof.* The proof follows by Theorem 2.1 with $\mathbf{M} = \mathbf{G}_c^b$, then bounding $\|\mathbf{u}\|_{\mathbf{G}_c^b}^2$ and $\|\mathbf{v}_t\|_{\mathbf{G}_c^b}^2$ via Lemma 4.1, and then using $\max_{t \in \mathcal{M}_A} \|\mathbf{v}_{i_t}\|_{\mathbf{G}}^2 \leq R_{\mathbf{G}}$ by equation (9). $\qquad\qquad\qquad\qquad\qquad\square$

The upper bound of the theorem is more resilient to label imbalance, concept noise, and label noise than the bound in [8, Theorems 3.2, 4.1, and 4.2] (see equation (1)). For example, given the noisy barbell graph in Figure 3 but with $k \ll n$ noisy vertices the bound (1) is $O(kn)$ while the bound (15) with $b = 1$, $c = 1$, and $|\mathcal{M}_{\mathbf{u}}| = 0$ is $O(k)$. A similar argument may be given for label imbalance.

In the bound above, for easy interpretability, one may upper bound the resistance diameter $R_{\mathbf{G}}$ by the geodesic diameter $D_{\mathbf{G}}$. However, the resistance diameter makes for a sharper bound in a number of natural situations. For example now consider (a thick barbell) two $m$-cliques (one labeled "+1", one "-1") with $\ell$ edges ($\ell < m$) between the cliques. We observe between any two vertices there are at least $\ell$ edge-disjoint paths of length no more than five, therefore the resistance diameter is at most $5/\ell$ by the "resistors-in-parallel" rule while the geodesic diameter is 3. Thus, for "thick barbells" if we use the geodesic diameter we have a mistake bound of $16\ell$ (substituting $\beta_{\mathbf{u}} = 0$, and $R_{\mathbf{G}} \leq 3$ into (16)) while surprisingly with the resistance diameter the bound (substituting $b = \frac{1}{4n}$, $c = 0$, $|\mathcal{M}_{\mathbf{u}}| = 0$, $\beta_{\mathbf{u}} = 0$, and $R_{\mathbf{G}} \leq 5/\ell$ into (15)) is independent of $\ell$ and is 20.

## 5    Discussion

In this paper, we have provided a bound on the performance of the perceptron on the graph in terms of structural properties of the graph and its labeling which are only indirectly dependent on the number of vertices in the graph, in particular, they depend on the cut size and the diameter. In the following, we compare the perceptron with two other approaches. First, we compare the percep-tron with the graph kernel $\mathbf{K}_0^1$ to the conceptually simpler $k$-nearest neighbors algorithm with either the graph geodesic distance or the resistance distance. In particular, we prove the impossibility of bounding performance of $k$-nearest neighbors only in terms of the diameter and the cut size. Specif-ically, we give a parameterized family of graphs for which the number of mistakes of the perceptron is upper bounded by a fixed constant independent of the graph size while $k$-nearest neighbors prov-ably incurs mistakes linearly in the graph size. Second, we compare the perceptron with the graph kernel $\mathbf{K}_0^1$ with a simple application of the classical *halving* algorithm [11]. Here, we conclude that the upper bound for the perceptron is better for graphs with a small diameter while the halving al-gorithm's upper bound is better for graphs with a large diameter. In the following, for simplicity we limit our discussion to binary-weighted graphs, noise-free data (see equation (16)) and upper bound the resistance diameter $R_{\mathbf{G}}$ with the geodesic diameter $D_{\mathbf{G}}$ (see equation (9)).

### 5.1    *K*-nearest neighbors on the graph

We consider the $k$-nearest neighbors algorithms on the graph with both the resistance distance (see equation (7)) and the graph geodesic distance. The geodesic distance between two vertices is the length of the shortest path between the two vertices (recall the discussion in section 3). In the following, we use the emphasis *distance* to refer simultaneously to both distances. Now, consider the family of $\mathcal{O}_{\ell,m,p}$ of octopus graphs. An octopus graph (see Figure 5) consists of a "head" which is an $\ell$-clique ($C^{(\ell)}$) with vertices denoted by $c_1, \ldots, c_\ell$, and a set of $m$ "tentacles" ($\{T_i\}_{i=1}^m$), where each tentacle is a line graph of length $p$. The vertices of tentacle $i$ are denoted by $\{t_{i,0}, t_{i,1}, \ldots, t_{i,p}\}$; the $t_{i,0}$ are all identified as one vertex $r$ which acts as the root of the $m$ tentacles. There is an edge (the body) connecting root $r$ to the vertex $c_1$ on the head. Thus, this graph has diameter $D = \max(p + 2, 2p)$ and there are $\ell + mp + 1$ vertices in total; an octopus $\mathcal{O}_{m,p}$ is balanced if $\ell = mp + 1$. Note that the *distance* of every vertex in the head to every other vertex in the graph is no more than $p + 2$, and every tentacle "tip" $t_{i,p}$ is *distance* $2p$ to other tips $t_{j,p} : j \neq i$.

We now argue that $k$-nearest neighbors may incur mistakes linear in the number of tentacles. To this end, choose $p \geq 3$ and suppose we have the following online data sequence

$$\{(c_1, +1), (t_{1,p}, -1), (c_2, +1), (t_{2,p}, -1), \ldots, (c_m, +1), (t_{m,p}, -1)\}.$$

Note that $k$-nearest neighbors will make a mistake on every instance $(t_{i,p}, -1)$ and so, even assuming that it predicts correctly on $(c_1, +1)$ it will always make $m$ mistakes. We now contrast this result with the performance of the perceptron with the graph kernel $\mathbf{K}_0^1$ (see equation (10)). By equation (16), the number of mistakes will be upper bounded by $10p + 5$ because there is a cut of size 1 and the

diameter is $2p$. Thus, for balanced octopi $\mathcal{O}_{m,p}$ with $p \geq 3$, as $m$ grows the number of mistakes of the kernel perceptron will be bounded by a fixed constant. Whereas *distance* $k$-nearest neighbors will incur mistakes linearly in $m$.

## 5.2  Halving algorithm

We now compare the performance of our algorithm to the classical *halving* algorithm [11]. The halving algorithm operates by predicting on each trial as the majority of the classifiers in the concept class which have been consistent over the trial sequence. Hence, the number of mistakes of the halving algorithm is upper bounded by the logarithm of the cardinality of the concept class. Let $\mathcal{K}_{\mathcal{G}}^k = \{\mathbf{u} \in \{-1,1\}^n : \Phi_{\mathbf{G}}(\mathbf{u}) = k\}$ be the set all of all classifiers with a cut size equal to $k$ on a fixed graph $\mathcal{G}$. The cardinality of $\mathcal{K}_{\mathcal{G}}^k$ is upper bounded by $\binom{n(n-1)}{k}$ since any classifier (cut) in $\mathcal{K}_{\mathcal{G}}^k$ can be uniquely identified by a choice of $k$ edges and 1 bit which determines the sign of the vertices in the same of partition (however we overcount as not every set of edges determines a classifier). The number of mistakes of the halving algorithm is upper bounded by $O(k \log \frac{n}{k})$. For example, on a line graph with a cut size of 1 the halving algorithm has an upper bound of $\lceil \log n \rceil$ while the upper bound for the number of mistakes of the perceptron as given in equation (16) is $5n + 5$. Although the halving algorithm has a sharper bound on such large diameter graphs as the line graph, it unfortunately has a logarithmic dependence on $n$. This contrasts to the bound of the perceptron which is essentially independent of $n$. Thus, the bound for the halving algorithm is roughly sharper on graphs with a diameter $\omega(\log \frac{n}{k})$, while the perceptron bound is roughly sharper on graphs with a diameter $o(\log \frac{n}{k})$. We emphasize that this analysis of upper bounds is quite rough and sharper bounds for both algorithms could be obtained for example, by including a term representing the minimal possible cut, that is, the minimum number of edges necessary to disconnect a graph. For the halving algorithm this would enable a better bound on the cardinality of $\mathcal{K}_{\mathcal{G}}^k$ (see [13]). While, for the perceptron the larger the connectivity of the graph, the weaker the diameter upper bound in Theorem 3.2 (see for example the discussion of "thick barbells" at the end of section 4).

### Acknowledgments

We wish to thank the anonymous reviewers for their useful comments. This work was supported by EPSRC Grant GR/T18707/01 and by the IST Programme of the European Community, under the PASCAL Network of Excellence IST-2002-506778.

## Footnotes

[1]Later in the paper we extend the definition of cut size to weighted graphs.

# References

[1] C. Gentile. The robustness of the p-norm algorithms. *Machine Learning*, 53(3):265–299, 2003.

[2] A. Blum and S. Chawla. Learning from labeled and unlabeled data using graph mincuts. In *ICML 2002*, pages 19–26. Morgan Kaufmann, San Francisco, CA, 2002.

[3] R. I. Kondor and J. Lafferty. Diffusion kernels on graphs and other discrete input spaces. In *ICML 2002*, pages 315–322. Morgan Kaufmann, San Francisco, CA, 2002.

[4] X. Zhu, Z. Ghahramani, and J. Lafferty. Semi-supervised learning using gaussian fields and harmonic functions. In *ICML 2003*, pages 912–919, 2003.

[5] A. Smola and R.I. Kondor. Kernels and regularization on graphs. In *COLT 2003*, pages 144–158, 2003.

[6] M. Belkin, I. Matveeva, and P. Niyogi. Regularization and semi-supervised learning on large graphs. In *COLT 2004*, pages 624 – 638, Banff, Alberta, 2004. Springer.

[7] T. Zhang and R. Ando. Analysis of spectral kernel design based semi-supervised learning. In Y. Weiss, B. Schölkopf, and J. Platt, editors, *NIPS 18*, pages 1601–1608. MIT Press, Cambridge, MA, 2006.

[8] M. Herbster, M. Pontil, and L. Wainer. Online learning over graphs. In *ICML 2005*, pages 305–312, New York, NY, USA, 2005. ACM Press.

[9] Y. Freund and R. E. Schapire. Large margin classification using the perceptron algorithm. *Machine Learning*, 37(3):277–296, 1999.

[10] D. Klein and M. Randić. Resistance distance. *Journal of Mathematical Chemistry*, 12(1):81–95, 1993.

[11] J. M. Barzdin and R. V. Frievald. On the prediction of general recursive functions. *Soviet Math. Doklady*, 13:1224–1228, 1972.

[12] N. Aronszajn. Theory of reproducing kernels. *Trans. Amer. Math. Soc.*, 68:337–404, 1950.

[13] D. Karger and C. Stein. A new approach to the minimum cut problem. *JACM*, 43(4):601–640, 1996.
